# Fully Bayesian inference for neural models with negative-binomial spiking

**Jonathan W. Pillow**
Center for Perceptual Systems
Department of Psychology
The University of Texas at Austin
pillow@mail.utexas.edu

**James G. Scott**
Division of Statistics and Scientific Computation
McCombs School of Business
The University of Texas at Austin
james.scott@mccombs.utexas.edu

## Abstract

Characterizing the information carried by neural populations in the brain requires accurate statistical models of neural spike responses. The negative-binomial distribution provides a convenient model for over-dispersed spike counts, that is, responses with greater-than-Poisson variability. Here we describe a powerful data-augmentation framework for fully Bayesian inference in neural models with negative-binomial spiking. Our approach relies on a recently described latent-variable representation of the negative-binomial distribution, which equates it to a Polya-gamma mixture of normals. This framework provides a tractable, conditionally Gaussian representation of the posterior that can be used to design efficient EM and Gibbs sampling based algorithms for inference in regression and dynamic factor models. We apply the model to neural data from primate retina and show that it substantially outperforms Poisson regression on held-out data, and reveals latent structure underlying spike count correlations in simultaneously recorded spike trains.

## 1 Introduction

A central problem in systems neuroscience is to understand the probabilistic representation of information by neurons and neural populations. Statistical models play a critical role in this endeavor, as they provide essential tools for quantifying the stochasticity of neural responses and the information they carry about various sensory and behavioral quantities of interest.

Poisson and conditionally Poisson models feature prominently in systems neuroscience, as they provide a convenient and tractable description of spike counts governed by an underlying spike rate. However, Poisson models are limited by the fact that they constrain the ratio between the spike count mean and variance to one. This assumption does not hold in many brain areas, particularly cortex, where responses are often over-dispersed relative to Poisson [1].

A second limitation of Poisson models in regression analyses (for relating spike responses to stimuli) or latent factor analyses (for finding common sources of underlying variability) is the difficulty of performing fully Bayesian inference. The posterior formed under Poisson likelihood and Gaussian prior has no tractable representation, so most theorists resort to either fast, approximate methods based on Gaussians, [2–9] or slower, sampling-based methods that may scale poorly with data or dimensionality [10–15].

The negative-binomial (NB) distribution generalizes the Poisson with a shape parameter that controls the tradeoff between mean and variance, providing an attractive alternative for over-dispersed spike count data. Although well-known in statistics, it has only recently been applied for neural data [16–18]. Here we describe fully Bayesian inference methods for the neural spike count data based on a recently developed representation of the NB as a Gaussian mixture model [19]. In the

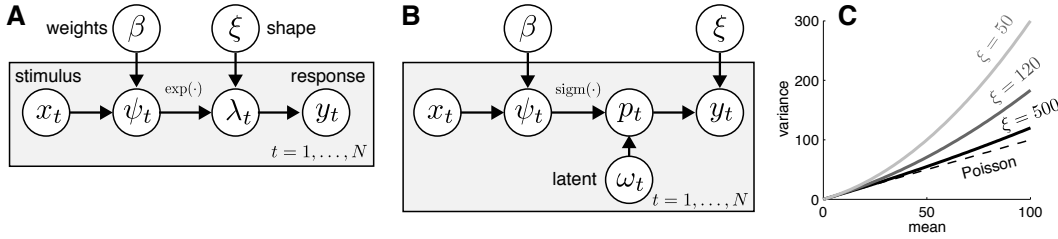

Figure 1: Representations of the negative-binomial (NB) regression model. **(A)** Graphical model for standard gamma-Poisson mixture representation of the NB. The linearly projected stimulus $\psi_t = \beta^T x_t$ defines the scale parameter for a gamma r.v. with shape parameter $\xi$, giving $\lambda_t \sim \text{Ga}(e^{\psi_t}, \xi)$, which is in turn the rate for a Poisson spike count: $y_t \sim \text{Poiss}(\lambda_t)$. **(B)** Graphical model illustrating novel representation as a Polya-Gamma (PG) mixture of normals. Spike counts are represented as NB distributed with shape $\xi$ and rate $p_t = 1/(1 + e^{-\psi_t})$. The latent variable $\omega_t$ is conditionally PG, while $\psi$ (and $\beta|x$) are normal given $(\omega_t, \xi)$, which facilitates efficient inference. **(C)** Relationship between spike-count mean and variance for different settings of shape parameter $\xi$, illustrating super-Poisson variability of the NB model.

following, we review the conditionally Gaussian representation for the negative-binomial (Sec. 2), describe batch-EM, online-EM and Gibbs-sampling based inference methods for NB regression (Sec. 3), sampling-based methods for dynamic latent factor models (Sec. 4), and show applications to spiking data from primate retina.

## 2   The negative-binomial model

Begin with the single-variable case where the data $Y = \{y_t\}$ are scalar counts observed at times $t = 1, \ldots, N$. A standard Poisson generalized linear model (GLM) assumes that $y_t \sim \text{Pois}(e^{\psi_t})$, where the log rate parameter $\psi_t$ may depend upon the stimulus. One difficulty with this model is that the variance of the Poisson distribution is equal to its mean, an assumption that is violated in many data sets [20–22].

To relax this assumption, we can consider the negative binomial model, which can be described as a doubly-stochastic or hierarchical Poisson model [18]. Suppose that $y_t$ arises according to:

$$
\begin{aligned}
(y_t \mid \lambda_t) &\sim \text{Pois}(\lambda_t) \\
(\lambda_t \mid \xi, \psi_t) &\sim \text{Ga}\left(\xi, e^{\psi_t}\right),
\end{aligned}
$$

where we have parametrized the Gamma distribution in terms of its shape and scale parameters. By marginalizing over the top-level model for $\lambda_t$, we recover a negative-binomial distribution for $y_t$:

$$
p(y_t \mid \xi, \psi_t) \propto (1 - p_t)^\xi \, p_t^{y_t},
$$

where $p_t$ is related to $\psi_t$ via the logistic transformation:

$$
p_t = \frac{e^{\psi_t}}{1 + e^{\psi_t}}.
$$

The extra parameter $\xi$ therefore allows for over-dispersion compared to the Poisson, with the count $y_t$ having expected value $\xi e^{\psi_t}$ and variance $\xi e^{\psi_t}(1 + e^{\psi_t})$. (See Fig. 1).

Bayesian inference for models of this form has long been recognized as a challenging problem, due to the analytically inconvenient form of the likelihood function. To see the difficulty, suppose that $\psi_t = x_t^T \beta$ is a linear function of known inputs $x_t = (x_{t1}, \ldots, x_{tP})^T$. Then the conditional posterior distribution for $\beta$, up to a multiplicative constant, is

$$
p(\beta \mid \xi, Y) \propto p(\beta) \cdot \prod_{t=1}^{N} \frac{\{\exp(x_t^T \beta)\}^{y_t}}{\{1 + \exp(x_t^T \beta)\}^{\xi + y_t}}, \tag{1}
$$

where $p(\beta)$ is the prior distribution, and where we have assumed for the moment that $\xi$ is fixed. The two major issues are the same as those that arise in Bayesian logistic regression: the response

depends non-linearly upon the parameters, and there is no natural conjugate prior $p(\beta)$ to facilitate posterior computation.

One traditional approach for Bayesian inference in logistic models is to work directly with the discrete-data likelihood. A variety of tactics along these lines have been proposed, including numerical integration [23], analytic approximations to the likelihood [24–26], or Metropolis-Hastings [27]. A second approach is to assume that the discrete outcome is some function of an unobserved continuous quantity or latent variable. This is most familiar in the case of Bayesian inference for the probit or dichotomized-Gaussian model [28, 29], where binary outcomes $y_i$ are assumed to be thresholded versions of a latent Gaussian quantity $z_i$. The same approach has also been applied to logistic and Poisson regression [30, e.g.]. Unfortunately, none of these schemes lead to a fully automatic approach to posterior inference, as they require either approximations (whose quality must be validated) or the careful selection of tuning constants (as is typically required when using, for example, the Metropolis–Hastings sampler in very high dimensions).

To proceed with Bayesian inference in the negative-binomial model, we appeal to a recent latent-variable construction (depicted in Fig. 1B) from [19] based on the theory of Polya-Gamma random variables. The basic result we exploit is that the negative binomial likelihood can be represented as a mixture of normals with Polya-Gamma mixing distribution. The algorithms that result from this scheme are both exact (in the sense of avoiding analytic approximations) and fully automatic.

**Definition 1.** *A random variable $X$ has a Polya-Gamma distribution with parameters $b > 0$ and $c \in \mathbb{R}$, denoted $X \sim PG(b, c)$, if*

$$X \overset{D}{=} \frac{1}{2\pi^2} \sum_{k=1}^{\infty} \frac{g_k}{(k - 1/2)^2 + c^2/(4\pi^2)} \,, \tag{2}$$

*where each $g_k \sim Ga(b, 1)$ is an independent gamma random variable, and where $\overset{D}{=}$ denotes equality in distribution.*

We make use of four important facts about Polya-Gamma variables from [19]. First, suppose that $p(\omega)$ denotes the density of the random variable $\omega \sim PG(b, 0)$, for $b > 0$. Then for any choice of $a$,

$$\frac{(e^\psi)^a}{(1 + e^\psi)^b} = 2^{-b} e^{\kappa\psi} \int_0^\infty e^{-\omega\psi^2/2} \, p(\omega) \, d\omega \,, \tag{3}$$

where $\kappa = a - b/2$. This integral identity allows us to rewrite each term in the negative binomial likelihood (eq. 1) as

$$(1 - p_t)^\xi \, p_t^{y_t} = \frac{\{\exp(\psi_t)\}^{y_t}}{\{1 + \exp(\psi_t)\}^{h+y_t}} \propto e^{\kappa_t \psi_t} \int_0^\infty e^{-\omega_t \psi^2/2} \, p(\omega \mid \xi + y_t, 0) \, d\omega \,, \tag{4}$$

where $\kappa_t = (y_t - \xi)/2$, and where the mixing distribution is Polya-Gamma. Conditional upon $\omega_t$, we have a likelihood proportional to $e^{-Q(\psi_t)}$ for some quadratic form $Q$, which will be conditionally conjugate to any Gaussian or mixture-of-Gaussians prior for $\psi_t$. This conditional Gaussianity can be exploited to great effect in MCMC, EM, and sequential Monte Carlo algorithms, as described in the next section.

A second important fact is that the conditional distribution

$$p(\omega \mid \psi) = \frac{e^{-\omega\psi^2/2} \, p(\omega)}{\int_0^\infty e^{-\omega\psi^2/2} \, p(\omega) \, d\omega}$$

is also in the Polya-Gamma class: $(\omega \mid \psi) \sim PG(b, \psi)$. In this sense, the Polya-Gamma distribution is conditionally conjugate to the NB likelihood, which is very useful for Gibbs sampling.

Third, although the density of a Polya-Gamma random variable can be expressed only as an infinite series, its expected value is known in closed form: if $\omega \sim PG(b, c)$, then

$$E(\omega) = \frac{b}{2c} \tanh(c/2) \,. \tag{5}$$

As we show in the next section, this expression comes up repeatedly when fitting negative-binomial models via expectation-maximization, where these moments of $\omega_t$ form a set of sufficient statistics for the complete-data log posterior distribution in $\beta$.

Finally, despite the awkward form of the density function, it is still relatively easy to simulate random Polya-Gamma draws, avoiding entirely the need to truncate the infinite sum in Equation 2. As the authors of [19] show, this can be accomplished via a highly efficient accept-reject algorithm using ideas from [31]. The proposal distribution requires only exponential, uniform, and normal random variates; and the algorithm's acceptance probability is uniformly bounded below at 0.9992 (implying roughly 8 rejected draws out of every 10,000 proposals).

As we now describe, these four facts are sufficient to allow straightforward Bayesian inference for negative-binomial models. We focus first on regression models, for which we derive simple Gibbs sampling and EM algorithms. We then turn to negative-binomial dynamic factor models, which can be fit using a variant of the forward-filter, backwards-sample (FFBS) algorithm [32].

## 3 Negative-binomial regression

### 3.1 Fully Bayes inference via MCMC

Suppose that $\psi_t = x_t^T \beta$ for some $p$-vector of regressors $x_t$. Then, conditional upon $\omega_t$, the contribution of observation $t$ to the likelihood is

$$
\begin{aligned}
L_t(\beta) &\propto \exp\{\kappa_t x_t^T \beta - \omega_t (x_t^T \beta)^2 / 2\} \\
&\propto \exp\left\{ -\frac{\omega_t}{2} \left( \frac{y_t - \xi}{2\omega_t} - x_t^T \beta \right)^2 \right\} .
\end{aligned}
$$

Let $\Omega = \mathrm{diag}(\omega_1, \ldots, \omega_n)$; let $z_t = (y_t - \xi)/(2\omega_t)$; and let $z$ denote the stacked vector of $z_t$ terms. Combining all terms in the likelihood leads to a Gaussian linear-regression model where

$$
(z \mid \beta, \Omega) \sim N(X\beta, \Omega^{-1}) .
$$

It is usually reasonable to assume a conditionally Gaussian prior, $\beta \sim N(c, C)$. Note that $C$ itself may be random, as in, for example, a Bayesian lasso or horseshoe prior [33–35]. Gibbs sampling proceeds in two simple steps:

$$
\begin{aligned}
(\omega_t \mid \xi, \beta) &\sim \mathrm{PG}(y_t + \xi, x_t^T \beta) \\
(\beta \mid \Omega, z) &\sim N(m, V) ,
\end{aligned}
$$

where PG denotes a Polya-Gamma draw, and where

$$
\begin{aligned}
V &= (X^T \Omega X + C^{-1})^{-1} \\
m &= V(X^T \Omega z + C^{-1} c) .
\end{aligned}
$$

One may update the dispersion parameter $\xi$ via Gibbs sampling, using the method described in [36].

### 3.2 Batch EM for MAP estimation

We may also use the same data-augmentation trick in an expectation-maximization (EM) algorithm to compute the maximum a-posteriori (MAP) estimate $\hat{\beta}$. Returning to the likelihood in (4) and ignoring constants of proportionality, we may write the complete-data log posterior distribution, given $\omega_1, \ldots, \omega_N$, as

$$
Q(\beta) = \log p(\beta \mid Y, \omega_1, \ldots, \omega_N) = \sum_{t=1}^{N} \left\{ (x_t^T \beta) \cdot \frac{y_t - \xi}{2} - \omega_t \frac{(x_t^T \beta)^2}{2} \right\} + \log p(\beta)
$$

for some prior $p(\beta)$. This expression is linear in $\omega_t$. Therefore we may compute $E\{Q(\beta)\}$ by substituting $\hat{\omega}_t = E(\omega_t \mid \beta)$, given the current value of $\beta$, into the above expression. Appealing to (5), these conditional expectations are available in closed form:

$$
E(\omega_t \mid \beta) = \left( \frac{\kappa_t}{x_t^T \beta} \right) \tanh(x_t^T \beta / 2) ,
$$

where $\kappa_t = (y_t - \xi)/2$. In the M step, we re-express $E\{Q(\beta)\}$ as

$$
E\{Q(\beta)\} = -\frac{1}{2} \beta^T S \beta + \beta^T d + \log p(\beta) ,
$$

where the complete-data sufficient statistics are

$$
\begin{aligned}
S &= X^T \hat{\Omega} X \\
d &= X^T \kappa
\end{aligned}
$$

for $\hat{\Omega} = \mathrm{diag}(\hat{\omega}_1, \ldots, \hat{\omega}_N)$ and $\kappa = (\kappa_1, \ldots, \kappa_N)^T$. Thus the M step is a penalized weighted least squares problem, which can be solved using standard methods. In fact, it is typically unnecessary to maximize $E\{Q(\beta)\}$ exactly at each iteration. As is well established in the literature on the EM algorithm, it is sufficient to move to a value of $\beta$ that merely improves that observed-data objective function. We have found that it is much faster to take a single step of the conjugate conjugate-gradient algorithm (in which case in will be important to check for improvement over the previous iteration); see, e.g. [37] for details.

### 3.3 Online EM

For very large data sets, the above batch algorithm may be too slow. In such cases, we recommend computing the MAP estimate via an online EM algorithm [38], as follows. Suppose that our current estimate of the parameter is $\beta^{(t-1)}$, and that the current estimate of the complete-data log posterior is

$$
Q(\beta) = -\frac{1}{2}\beta^T S^{(t-1)} \beta + \beta^T d^{(t-1)} + \log p(\beta) , \tag{6}
$$

where

$$
\begin{aligned}
S^{(t-1)} &= \sum_{i=1}^{t-1} \hat{\omega}_i x_i x_i^T \\
d^{(t-1)} &= \sum_{i=1}^{t-1} \kappa_i x_i ,
\end{aligned}
$$

recalling that $\kappa_i = (y_i - \xi)/2$. After observing new data $(y_t, x_t)$, we first compute the expected value of $\omega_t$ as

$$
\hat{\omega}_t = E(\omega_t \mid y_t, \beta^{(t-1)}) = \left(\frac{\kappa_t}{\psi_t}\right) \tanh(\psi_t/2) ,
$$

with $\psi_t = x_t^T \beta^{(t-1)}$ denoting the linear predictor evaluated at the current estimate. We then update the sufficient statistics recursively as

$$
\begin{aligned}
S^{(t)} &= (1 - \gamma_t) S^{(t-1)} + \gamma_t \hat{\omega}_t x_t x_t^T \\
d^{(t)} &= (1 - \gamma_t) d^{(t-1)} + \gamma_t \kappa_t x_t ,
\end{aligned}
$$

where $\gamma_t$ is the learning rate. We then plug these updated sufficient statistics into (6), and solve the M step to move to a new value of $\beta$. The data can also be processed in batches of size larger than 1, with obvious modifications to the updates for $S^{(t)}$ and $d^{(t)}$; we have found that batch sizes of order $\sqrt{p}$ tend to work well, although we are unaware of any theory to support this choice.

In high-dimensional problems, the usual practice is to impose sparsity via an $\ell^1$ penalty on the regression coefficients, leading to a lasso-type prior. In this case, the M-step in the online algorithm can be solved very efficiently using the modified shooting algorithm, a coordinate-descent method described in a different context by [39] and [40].

This online EM is guaranteed to converge to a stationary point of the log posterior distribution if the learning rate decays in time such that $\sum_{t=1}^{\infty} \gamma_t = \infty$ and $\sum_{t=1}^{\infty} \gamma_t^2 < \infty$. (If the penalty function is concave and $\xi$ is fixed, then this stationary point will be the global maximum.) A simple choice for the learning rate is $\gamma_t = 1/t^a$ for $a \in (0.5, 1)$, with $a = 0.7$ being our default choice.

## 4 Factor analysis for negative-binomial spiking

Let $\psi_t = (\psi_{t1}, \ldots, \psi_{tK})$ denote a vector of $K$ linear predictors at time $t$, corresponding to $K$ different neurons with observed counts $Y_t = (y_{t1}, \ldots, y_{tK})^T$. We propose a dynamic negative-

binomial factor model for $Y_t$, with a vector autoregressive (VAR) structure for the latent factors:

$$
\begin{aligned}
y_{tk} &\sim \mathrm{NB}(\xi, e^{\psi_{tk}}) \quad \text{for} \quad k = 1, \dots K \\
\psi_t &= \alpha + B f_t \\
f_t &= \Gamma f_{t-1} + \epsilon_t \,, \quad \epsilon_t \sim \mathrm{N}(0, \tau^2 I) \,.
\end{aligned}
$$

Here $f_t$ denotes an $L$-vector of latent factors, with $L$ typically much smaller than $P$. The $K \times L$ factor-loadings matrix $B$ is restricted to have zeroes above the diagonal, and to have positive diagonal entries. These restrictions are traditional in Bayesian factor analysis [41], and ensure that $B$ is formally identified. We also assume that $\Gamma$ is a diagonal matrix, and impose conjugate inverse-gamma priors on $\tau^2$ to ensure that, marginally over the latent factors $f_t$, the entries of $\psi_t$ have approximately unit variance. Although we do not pursue the point here, the mean term $\alpha$ can incorporate the effect of known predictors with no additional complication to the analysis.

By exploiting the Polya-Gamma data-augmentation scheme, posterior inference in this model may proceed via straightforward Gibbs sampling—something not previously possible for count-data factor models. Prior work on latent variable modeling of spike data has relied on either Gaussian approximations [2–6, 8] or variants of particle filtering [10–13].

Gibbs sampling proceeds as follows. Conditional upon $B$ and $f_t$, we update the latent variables as $\omega_{tk} \sim \mathrm{PG}(y_{tk} + \xi, B_k f_t)$, where $B_k$ denotes the $k$th row of the loadings matrix. The mean vector $\alpha$ and factor-loadings matrix $B$ can both be updated in closed-form via a Gaussian draw using the full conditional distributions given in, for example, [42] or [43].

Given all latent variables and other parameters of the model, the factors $f_t$ can be updated in a single block using the forward-filter, backwards-sample (FFBS) algorithm from [32]. First, pass forwards through the data from $y_1$ to $y_N$, recursively computing the filtered moments of $f_t$ as

$$
\begin{aligned}
M_t &= (V_t^{-1} + B^T \Omega_t B)^{-1} \\
m_t &= M_t (B^T \Omega_t z_t + V_t^{-1} \Gamma m_{t-1}) \,,
\end{aligned}
$$

where

$$
\begin{aligned}
V_t &= \Gamma M_{t-1} \Gamma^T + \tau^2 I \\
z_t &= (z_{t1}, \dots, z_{tK})^T \quad , \quad z_{tk} = \frac{y_{tk} - \xi}{2 \omega_{tk}} - \alpha_k \\
\Omega_t &= \mathrm{diag}(\omega_{t1}, \dots, \omega_{tK}) \,.
\end{aligned}
$$

Then draw $f_N \sim \mathrm{N}(m_N, M_N)$ from its conditional distribution. Finally, pass backwards through the data, sampling $f_t$ as $(f_t \mid m_t, M_t, f_{t+1}) \sim \mathrm{N}(a_t, A_t)$, where

$$
\begin{aligned}
A_t^{-1} &= M_t^{-1} + \tau^{-2} I \\
a_t &= A_t^{-1}(M_t^{-1} m_t + \tau^{-2} f_{t+1}) \,.
\end{aligned}
$$

This will result in a block draw of all $N \times L$ factors from their joint conditional distribution.

## 5 Experiments

To demonstrate our methods, we performed regression and dynamic factor analyses on a dataset of 27 neurons recorded from primate retina (published in [44] and re-used with authors' permission). Briefly, these data consist of spike responses from a simultaneously-recorded population of ON and OFF parasol retinal ganglion cells, stimulated with a flickering, 120-Hz binary white noise stimulus.

### 5.1 Regression

Figure 2 shows a comparison of a Poisson model versus a negative-binomial model for each of the 27 neurons in the retinal dataset. We binned spike counts in 8 ms bins, and regressed against a temporally lagged stimulus, resulting in a 100-element ($10 \times 10$ pixel) spatial receptive field $\beta$ for each neuron. To benchmark the two methods, we created 50 random train/test splits from a full dataset of 30,000 points, with 7,500 points held out for validation. Using each training set, we used

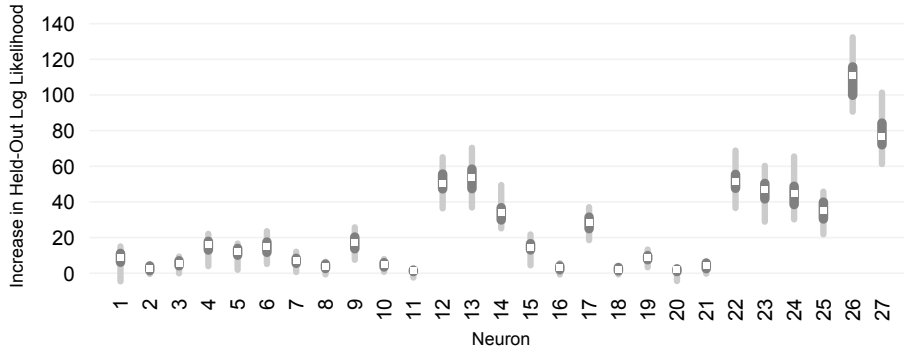

Figure 2: Boxplots of improvement in held-out log likelihoods (NB versus Poisson regression) for 50 train/test splits on each of the 27 neurons in the primate retinal data.

our online maximum-likelihood method to fit an NB model to each of the 27 neurons, and then used these models to compute held-out log-likelihoods on the test set versus a standard Poisson GLM. As Figure 2 shows, the NB model has a higher average held-out log-likelihood than the Poisson model. In some cases it is dozens of orders of magnitude better (as in neurons 12–14 and 22–27), suggesting that there is substantial over-dispersion in the data that is not faithfully captured by the Poisson model. We emphasize that this is a "weak-signal" regime, and that overdispersion is likely to be less when the signal is stronger. Yet these results suggest, at the very least, that many of these neurons have marginal distributions that are quite far from Poisson. Moreover, regardless of the underlying signal strength, the regression problem can be handled quite straightforwardly using our online method, even in high dimensions, without settling for the restrictive Poisson assumption.

## 5.2 Dynamic factor analysis

To study the factor-modeling framework, we conducted parallel experiments on both simulated and real data. First, we simulated two different data sets comprising 1000 time points and 11 neurons, each from a two-factor model: one with high factor autocorrelation ($\Gamma = 0.98$), and one with low factor autocorrelation ($\Gamma = 0.5$). The two questions of interest here are: how well does the fully Bayesian method reconstruct the correlation structure among the unobserved rate parameters $\psi_{tk}$; and how well does it distinguish between a high-autocorrelation and low-autocorrelation regime in the underlying low-dimensional representation?

The results in Figure 3 suggest that the results, on both counts, are highly accurate. It is especially interesting to compare the left-most column of Figure 3 with the actual cross-sectional correlation of $\psi_t$, the systematic component of variation, in the second column. The correlation of the raw counts $y_t$ show a dramatic attenuation effect, compared to the real latent states. Yet this structure is uncovered easily by the model, with together with a full assessment of posterior uncertainty. The approach behaves much like a model-based version of principal-components analysis, appropriate for non-Gaussian data.

Finally, Figure 4 shows the results of fitting a two-factor model to the primate retinal data. We are able to uncover latent structure in the data in a completely unsupervised fashion. As with the simulated data, it is interesting to compare the correlation of the raw counts $y_t$ with the estimated correlation structure of the latent states. There is also strong support for a low-autocorrelation regime in the factors, in light of the posterior mean factor scores depicted in the right-most pane.

## 6 Discussion

Negative-binomial models have only recently been explored in systems neuroscience, despite their favorable properties for handling data with larger-than-Poisson variation. Likewise, Bayesian inference for the negative binomial model has traditionally been a difficult problem, with the existence of a fully automatic Gibbs sampler only recently discovered [19]. Our paper has made three specific contributions to this literature. First, we have shown that negative-binomial models can lead to

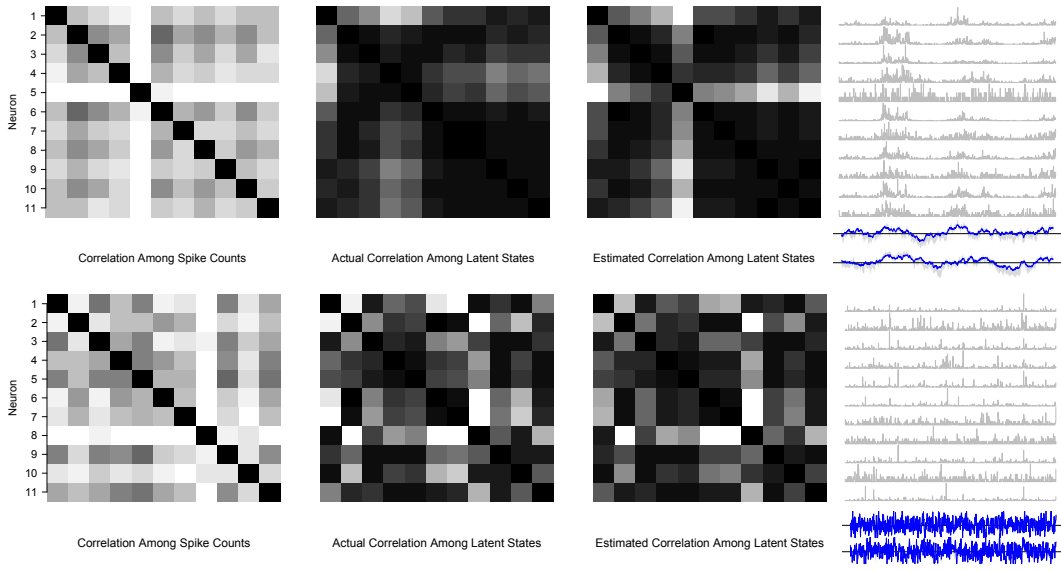

Figure 3: Results for two simulated data sets with high factor autocorrelation (top row) and low factor autocorrelation (bottom row). The three left-most columns show the raw correlation among the counts $y_t$; the actual correlation, $E(\psi_t \psi_t^T)$, of the latent states; and the posterior mean estimator for the correlation of the latent states. The right-most column shows the simulated spike trains for the 11 neurons, along with the factors $f_t$ in blue (with $75\%$ credible intervals), plotted over time.

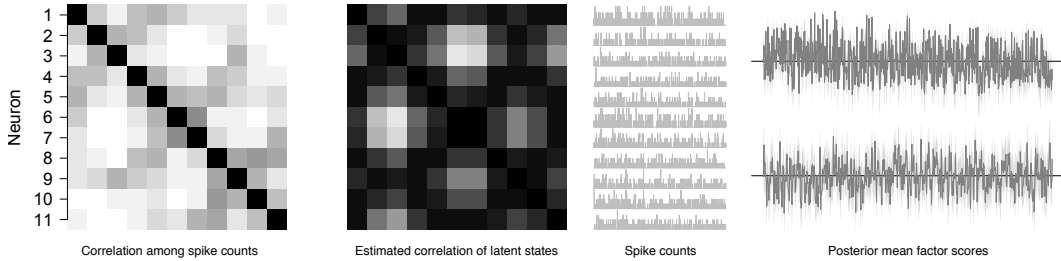

Figure 4: Results for factor analysis of the primate retinal data.

substantial improvements in fit, compared to the Poisson, for neural data exhibiting over-dispersion. Such models can be fit straightforwardly via MCMC for a wide class of prior distributions over model parameters (including sparsity-inducing choices, such as the lasso). Second, we have proposed a novel online-EM algorithm for sparse NB regression. This algorithm inherits all the convergence properties of EM, but is scalable to extremely large data sets. Finally, we have embedded a dynamic factor model inside a negative-binomial likelihood. This latter approach can be extended quite easily to spatial interactions, more general state-space models, or mixed models incorporating both regressors and latent variables. All of these extensions, as well as the model-selection question (how many factors?) form promising areas for future research.

## Acknowledgments

We thank E. J. Chichilnisky, A. M. Litke, A. Sher and J. Shlens for retinal data, J. Windle for PG sampling code, and J. H. Macke for helpful comments. This work was supported by a Sloan Research Fellowship, McKnight Scholar's Award, and NSF CAREER Award IIS-1150186 (JP).

# References

[1] Roland Baddeley, L. F. Abbott, Michael C. A. Booth, Frank Sengpiel, Tobe Freeman, Edward A. Wakeman, and Edmund T. Rolls. *Proceedings of the Royal Society of London. Series B: Biological Sciences*, 264(1389):1775–1783, 1997.

[2] E. Brown, L. Frank, D. Tang, M. Quirk, and M. Wilson. *Journal of Neuroscience*, 18:7411–7425, 1998.

[3] L. Srinivasan, U. Eden, A. Willsky, and E. Brown. *Neural Computation*, 18:2465–2494, 2006.

[4] B. M. Yu, J. P. Cunningham, G. Santhanam, S. I. Ryu, K. V. Shenoy, and M. Sahani. *Journal of Neurophysiology*, 102(1):614, 2009.

[5] W. Wu, J.E. Kulkarni, N.G. Hatsopoulos, and L. Paninski. *Neural Systems and Rehabilitation Engineering, IEEE Transactions on*, 17(4):370–378, 2009.

[6] Liam Paninski, Yashar Ahmadian, Daniel Gil Ferreira, Shinsuke Koyama, Kamiar Rahnama Rad, Michael Vidne, Joshua Vogelstein, and Wei Wu. *J Comput Neurosci*, Aug 2009.

[7] J. W. Pillow, Y. Ahmadian, and L. Paninski. *Neural Comput*, 23(1):1–45, Jan 2011.

[8] M Vidne, Y Ahmadian, J Shlens, J W Pillow, J Kulkarni, A M Litke, E J Chichilnisky, E P Simoncelli, and L Paninski. *J. Computational Neuroscience*, pages 1–25, 2012. To appear.

[9] John P. Cunningham, Krishna V. Shenoy, and Maneesh Sahani. *Proceedings of the 25th international conference on Machine learning*, ICML '08, pages 192–199, New York, NY, USA, 2008. ACM.

[10] A. E. Brockwell, A. L. Rojas, and R. E. Kass. *J Neurophysiol*, 91(4):1899–1907, Apr 2004.

[11] S. Shoham, L. Paninski, M. Fellows, N. Hatsopoulos, J. Donoghue, and R. Normann. *IEEE Transactions on Biomedical Engineering*, 52:1312–1322, 2005.

[12] Ayla Ergun, Riccardo Barbieri, Uri T. Eden, Matthew A. Wilson, and Emery N. Brown. *IEEE Trans Biomed Eng*, 54(3):419–428, Mar 2007.

[13] A. E. Brockwell, R. E. Kass, and A. B. Schwartz. *Proceedings of the IEEE*, 95:1–18, 2007.

[14] R. P. Adams, I. Murray, and D. J. C. MacKay. *Proceedings of the 26th Annual International Conference on Machine Learning*. ACM New York, NY, USA, 2009.

[15] Y. Ahmadian, J. W. Pillow, and L. Paninski. *Neural Comput*, 23(1):46–96, Jan 2011.

[16] M.C. Teich and W.J. McGill. *Physical Review Letters*, 36(13):754–758, 1976.

[17] Arno Onken, Steffen Grnewlder, Matthias H. J. Munk, and Klaus Obermayer. *PLoS Comput Biol*, 5(11):e1000577, 11 2009.

[18] R Goris, E P Simoncelli, and J A Movshon. *Computational and Systems Neuroscience (CoSyNe)*, Salt Lake City, Utah, February 2012.

[19] N.G. Polson, J.G. Scott, and J. Windle. *Arxiv preprint arXiv:1205.0310*, 2012.

[20] P. Lánskỳ and J. Vaillant. *Biosystems*, 58(1):27–32, 2000.

[21] V. Ventura, C. Cai, and R.E. Kass. *Journal of neurophysiology*, 94(4):2928–2939, 2005.

[22] *Neural Comput*, 18(11):2583–2591, Nov 2006.

[23] A.M. Skene and J. C. Wakefield. *Statistics in Medicine*, 9:919–29, 1990.

[24] J. Carlin. *Statistics in Medicine*, 11:141–58, 1992.

[25] Eric T. Bradlow, Bruce G. S. Hardie, and Peter S. Fader. *Journal of Computational and Graphical Statistics*, 11(1):189–201, 2002.

[26] A. Gelman, A. Jakulin, M.G. Pittau, and Y. Su. *The Annals of Applied Statistics*, 2(4):1360–83, 2008.

[27] A. Dobra, C. Tebaldi, and M. West. *Journal of Statistical Planning and Inference*, 136(2):355–72, 2006.

[28] James H. Albert and Siddhartha Chib. *Journal of the American Statistical Association*, 88(422):669–79, 1993.

[29] M. Bethge and P. Berens. *Advances in neural information processing systems*, 20:97–104, 2008.

[30] C. Holmes and L. Held. *Bayesian Analysis*, 1(1):145–68, 2006.

[31] Luc Devroye. *Statistics & Probability Letters*, 79(21):2251–9, 2009.

[32] Chris Carter and Robert Kohn. *Biometrika*, 81(541-53), 1994.

[33] Trevor Park and George Casella. *Journal of the American Statistical Association*, 103(482):681–6, 2008.

[34] Chris M. Hans. *Biometrika*, 96(4):835–45, 2009.

[35] Carlos M. Carvalho, Nicholas G. Polson, and James G. Scott. *Biometrika*, 97(2):465–80, 2010.

[36] Mingyuan Zhou, Lingbo Li, David Dunson, and Lawrence Carin. *International Conference on Machine Learning (ICML)*, 2012.

[37] Nicholas G. Polson and James G. Scott. Technical report, University of Texas at Austin, http://arxiv.org/abs/1103.5407v3, 2011.

[38] O. Cappé and E. Moulines. *Journal of the Royal Statistical Society (Series B)*, 71(3):593–613, 2009.

[39] Suhrid Balakrishnan and David Madigan. *Journal of Machine Learning Research*, 9:313–37, 2008.

[40] Liang Sun and James G. Scott. Technical report, University of Texas at Austin, 2012.

[41] H. Lopes and M. West. *Statistica Sinica*, 14:41–67, 2004.

[42] Joyee Ghosh and David B. Dunson. *Journal of Computational and Graphical Statistics*, 18(2):306–20, 2009.

[43] P.R. Hahn, Carlos M. Carvalho, and James G. Scott. *Journal of the Royal Statistical Society, Series C*, 2012.

[44] J. W. Pillow, J. Shlens, L. Paninski, A. Sher, A. M. Litke, and E. P. Chichilnisky, E. J. Simoncelli. *Nature*, 454:995–999, 2008.

